# Benchmarking Feed-Forward Neural Networks: Models and Measures

**Leonard G. C. Hamey**
Computing Discipline
Macquarie University
NSW 2109
AUSTRALIA

## Abstract

Existing metrics for the learning performance of feed-forward neural networks do not provide a satisfactory basis for comparison because the choice of the training epoch limit can determine the results of the comparison. I propose new metrics which have the desirable property of being independent of the training epoch limit. The *efficiency* measures the yield of correct networks in proportion to the training effort expended. The *optimal* epoch limit provides the greatest efficiency. The learning performance is modelled statistically, and asymptotic performance is estimated. Implementation details may be found in (Hamey, 1992).

## 1 Introduction

The empirical comparison of neural network training algorithms is of great value in the development of improved techniques and in algorithm selection for problem solving. In view of the great sensitivity of learning times to the random starting weights (Kolen and Pollack, 1990), individual trial times such as reported in (Rumelhart, *et al.*, 1986) are almost useless as measures of learning performance.

Benchmarking experiments normally involve many training trials (typically $N = 25$ or 100, although Tesauro and Janssens (1988) use $N = 10000$). For each trial $i$, the training time to obtain a correct network $t_i$ is recorded. Trials which are not successful within a limit of $T$ epochs are considered failures; they are recorded as $t_i = T$. The mean successful training time $\bar{t}_T$ is defined as follows.

$$\bar{t}_T = \frac{\sum_{t_i < T} t_i}{S}$$

where $S$ is the number of successful trials. The median successful time $\tilde{t}_T$ is the epoch at which $S/2$ trials are successes. It is common (e.g. Jacobs, 1987; Kruschke and Movellan, 1991; Veitch and Holmes, 1991) to report the mean and standard deviation along with the success rate $\lambda_T = S/N$, but the results are strongly dependent on the choice of $T$ as shown by Fahlman (1988). The problem is to characterise training performance independent of $T$.

Tesauro and Janssens (1988) use the harmonic mean $\bar{t}_H$ as the average learning rate.

$$\bar{t}_H = \frac{N}{\sum_{i=1}^{N} \frac{1}{t_i}}$$

This minimizes the contribution of large learning times, so changes in $T$ will have little effect on $\bar{t}_H$. However, $\bar{t}_H$ is not an unbiased estimator of the mean, and is strongly influenced by the shortest learning times, so that training algorithms which produce greater variation in the learning times are preferred by this measure.

Fahlman (1988) allows the learning program to restart an unsuccessful trial, incorporating the failed training time in the total time for that trial. This method is realistic, since a failed trial would be restarted in a problem-solving situation. However, Fahlman's averages are still highly dependent upon the epoch limit $T$ which is chosen beforehand as the restart point.

The present paper proposes new performance measures for feed-forward neural networks. In section 4, the optimal epoch limit $T_E$ is defined. $T_E$ is the optimal restart point for Fahlman's averages, and the efficiency $e$ is the scaled reciprocal of the optimised Fahlman average. In sections 5 and 6, the asymptotic learning behaviour is modelled and the mean and median are corrected for the truncation effect of the epoch limit $T$. Some benchmark results are presented in section 7, and compared with previously published results.

## 2    Performance Measurement

For benchmark results to be useful, the parameters and techniques of measurement and training must be fully specified. Training parameters include the network structure, the learning rate $\eta$, the momentum term $\alpha$ and the range of the initial random weights $[-r, r]$.

For problems with binary output, the correctness of the network response is defined by a threshold $\tau_c$—responses less than $\tau_c$ are considered equivalent to 0, while responses greater than $1 - \tau_c$ are considered equivalent to 1. For problems with analog output, the network response is considered correct if it lies within $\tau_c$ of the desired value. In the present paper, only binary problems are considered and the value $\tau_c = 0.4$ is used, as in (Fahlman 1988).

## 3    The Training Graph

The training graph displays the proportion of correct networks as a function of the epoch. Typically, the tail of the graph resembles a decay curve. It is evident in figure 1 that the

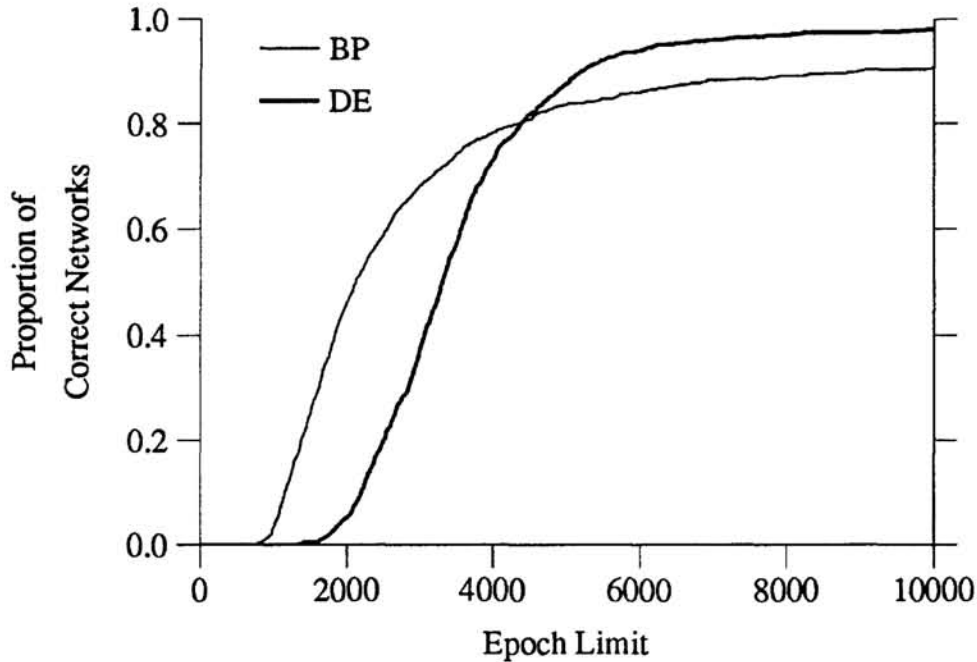

Figure 1: Typical Training Graphs: Back-Propagation ($\eta = 0.5, \alpha = 0$) and Descending Epsilon ($\eta = 0.5, \alpha = 0$) on Exclusive-Or (2–2–1 structure, $N = 1000, T = 10000$).

success rate for either algorithm may be significantly increased if the epoch limit was raised beyond 10000. The shape of the training graph varies depending upon the problem and the algorithm employed to solve it. Descending epsilon (Yu and Simmons, 1990) solves a higher proportion of the exclusive-or trials with $T = 10000$, but back-propagation would have a higher success rate if $T = 3000$. This exemplifies the dramatic effect that the choice of $T$ can have on the comparison of training algorithms.

Two questions naturally arise from this discussion: "What is the optimal value for $T$?" and "What happens as $T \rightarrow \infty$?". These questions will be addressed in the following sections.

## 4   Efficiency and Optimal $T$.

Adjusting the epoch limit $T$ in a learning algorithm affects both the yield of correct networks and the effort expended on unsuccessful trials. To capture the total yield for effort ratio, we define the efficiency $E(t)$ of epoch limit $t$ as follows.

$$E(t) = 1000 \frac{\sum_{t_i < t} 1}{\sum_{i=1}^{N} \min(t, t_i)}$$

The efficiency graph plots the efficiency against of the epoch limit. The efficiency graph for back-propagation (figure 2) exhibits a strong peak with the efficiency reducing relatively quickly if the epoch limit is too large. In contrast, the efficiency graph for descending epsilon exhibits an extremely broad peak with only a slight drop as the epoch limit is increased. This occurs because the asymptotic success rate ($\lambda$ in section 5) is close to

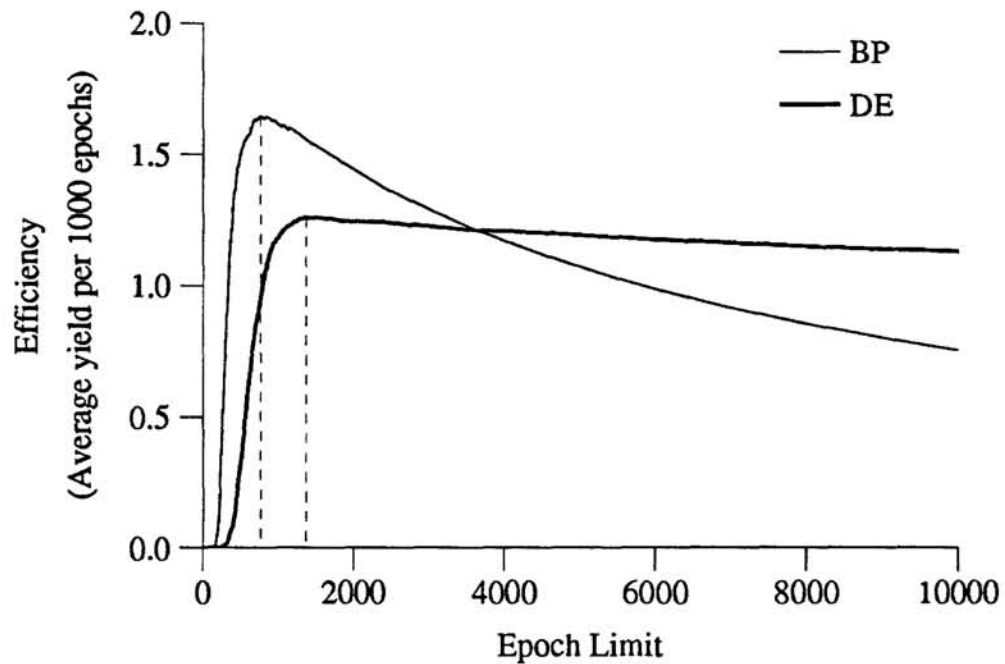

Figure 2: Efficiency Graphs: Back-Propagation ($\eta = 0.3, \alpha = 0.9$) and Descending Epsilon ($\eta = 0.3, \alpha = 0.9$) on Exclusive-Or (2–2–1 structure, $N = 1000, T = 10000$).

1.0; in such cases, the efficiency remains high over a wider range of epoch limits and near-optimal performance can be more easily achieved for novel problems.

The efficiency benchmark parameters are derived from the graph as shown in figure 3. The epoch limit $T_E$ at which the peak efficiency occurs is the optimal epoch limit. The peak efficiency $e$ is a good performance measure, independent of $T$ when $T > T_E$. Unlike $\bar{t}_H$, it is not biased by the shortest learning times. The peak efficiency is the scaled reciprocal of Fahlman's (1988) average for optimal $T$, and incorporates the failed trials as a performance penalty. The optimisation of training parameters is suggested by Tesauro and Janssens (1988), but they do not optimise $T$. For comparison with other performance measures, the unscaled optimised Fahlman average $\bar{t}_E = 1000/e$ may be used instead of $e$.

The prediction of the optimal epoch limit $T_E$ for novel problems would help reduce wasted computation. The range parameters $T_{E1}$ and $T_{E2}$ show how precisely $T$ must be set to obtain efficiency within 50% of optimal—if two algorithms are otherwise similar in performance, the one with a wider range $(T_{E1}, T_{E2})$ would be preferred for novel problems.

## 5    Asymptotic Performance: $T \to \infty$

In the training graph, the proportion of trials that ultimately learn correctly can be estimated by the asymptote which the graph is approaching. I statistically model the tail of the graph by the distribution $F(t) = 1 - [a(t - T_0) + 1]^{-k}$ and thus estimate the asymptotic success rate $\lambda$. Figure 4 illustrates the model parameters. Since the early portions of the graph are dominated by initialisation effects, $T_0$, the point where the model commences to fit, is determined by applying the Kolmogorov-Smirnov goodness-of-fit test (Stephens 1974)

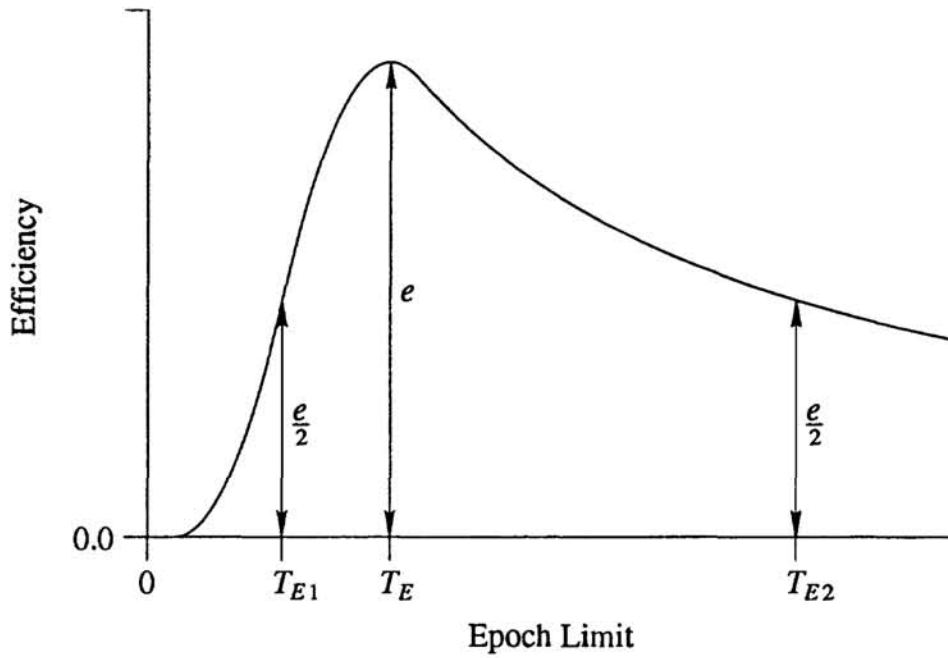

Figure 3: Efficiency Parameters in Relation to the Efficiency Graph.

for all possible values of $T_0$. The maximum likelihood estimates of $a$ and $k$ are found by using the simplex algorithm (Caceci and Cacheris, 1984) to directly maximise the following log-likelihood equation.

$$
\mathcal{L}(\mathbf{t}) = M \left[ \ln a + \ln k - \ln \left( 1 - \left( a(T - T_0) + 1 \right)^{-k} \right) \right] - \\
(k+1) \sum_{T_0 < t_i < T} \ln \left( a(t_i - T_0) + 1 \right)
$$

where $M$ is the number of trials recording times in the range $(T_0, T)$. The asymptotic success rate $\lambda$ is then obtained as follows.

$$
\lambda = \gamma + \frac{\lambda_T (1 - \gamma)}{F(T)}
$$

In practice, the statistical model I have chosen is not suitable for all learning algorithms. For example, in preliminary investigations I have been unable to reliably model the descending epsilon algorithm (Yu and Simmons, 1990). Further study is needed to develop more widely applicable models.

## 6   Corrected Measures

The mean $\bar{t}_T$ and the median $\tilde{t}_T$ are based upon only those trials that succeeded in $T$ epochs. The asymptotic learning model predicts additional success for $t > T$ epochs. Incorporating

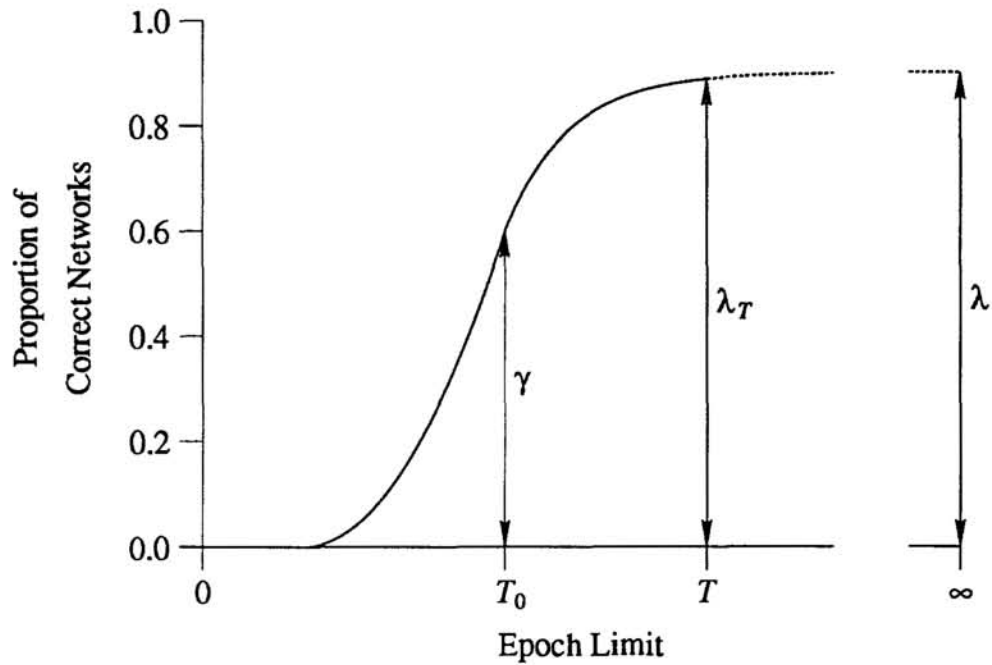

Figure 4: Parameters for the Model of Asymptotic Performance.

the predicted successes, the corrected mean $\bar{t}_C$ estimates the mean successful learning time as $T \to \infty$.

$$\bar{t}_C = \frac{\lambda_T \bar{t}_T + (\lambda - \lambda_T) \left(\frac{1}{a(k-1)} + T\right)}{\lambda}$$

The corrected median $\tilde{t}_C$ is the epoch for which $\lambda/2$ of the trials are successes. It estimates the median successful learning time as $T \to \infty$.

## 7    Benchmark Results for Back-Propagation

Table 1 presents optimised results for two popular benchmark problems: the 2–2–1 exclusive-or problem (Rumelhart, *et al.*, 1986, page 334), and the 10–5–10 encoder/decoder problem (Fahlman, 1988). Both problems employ three-layer networks with one hidden layer fully connected to the input and output units. The networks were trained with input and output values of 0 and 1. The weights were updated after each epoch of training; i.e. after each cycle through all the training patterns.

The characteristics of the learning for these two problems differs significantly. To accurately benchmark the exclusive-or problem, $N = 10000$ learning runs were needed to measure $e$ accurate to $\pm 0.3$. With $T = 200$, I searched the combinations of $\alpha$, $\eta$ and $r$. The optimal parameters were then used in a separate run with $N = 10000$ and $T = 2000$ to estimate the other benchmark parameters. In contrast, the encoder/decoder problem produced more stable efficiency values so that $N = 100$ learning runs produced estimates of $e$ precise to $\pm 0.2$. With $T = 600$, all the learning runs converged. The final benchmark values were

Table 1: Optimised Benchmark Results.

| PROBLEM | $r$ | $\alpha$ | $\eta$ | $e$ | $T_E$ | $T_{E1}$ | $T_{E2}$ | $\bar{t}_E$ |
|---|---|---|---|---|---|---|---|---|
| exclusive-or 2–2–1 | 1.4 ±0.2 | 0.65 ±0.05 | 7.0 ±0.5 | 17.1 ±0.3 | 49 | 26 | 235 | 59 |
| encoder/decoder 10–5–10 | 1.1 ±0.2 | 0.00 ±0.10 | 1.7 ±0.1 | 8.1 ±0.2 | $\infty$ | 110 | $\infty$ | 124 |

| PROBLEM | $a$ | $k$ | $T_0$ | $\gamma$ | $\lambda$ | $\bar{t}_C$ | $\lambda_T$ | $\bar{t}_T$ | $\bar{t}_H$ |
|---|---|---|---|---|---|---|---|---|---|
| exclusive-or | 0.1 | 0.5 | 54 | 0.66 | 0.93 | 409 | 0.76 | 50 | 40 |
| encoder/decoder | | | | | 1.00 | 124 | 1.00 | 124 | 114 |

determined with $N = 1000$. Confidence intervals for $e$ were obtained by applying the jackknife procedure (Mosteller and Tukey, 1977, chapter 8); confidence intervals on the training parameters reflect the range of near-optimal efficiency results.

In the exclusive-or results, the four means vary from each other considerably. $\bar{t}_C$ is large because the asymptotic performance model predicts many successful learning runs with $T > 2000$. However, since the model is fitting only a small portion of the data (approximately 1000 cases), its predictions may not be highly reliable. $\bar{t}_T$ is low because the limit $T = 2000$ discards the longer training runs. $\bar{t}_H$ is also low because it is strongly biased by the shortest times. $\bar{t}_E$ measures the training effort required per trained network, including failure times, provided that $T = 49$. However, $T_{E1}$ and $T_{E2}$ show that $T$ can lie within the range (26,235) and achieve performance no worse than 118 epochs effort per trained network.

The results for the encoder/decoder problem agree well with Fahlman (1988) who found $\alpha = 0$, $\eta = 1.7$ and $r = 1.0$ as optimal parameter values and obtained $\bar{t} = 129$ based upon $N = 25$. Equal performance is obtained with $\alpha = 0.1$ and $\eta = 1.6$, but momentum values in excess of 0.2 reduce the efficiency. Since all the learning runs are successful, $\bar{t}_E = \bar{t}_C = \bar{t}_T$ and $\lambda = \lambda_T = 1.0$. Both $T_E$ and $T_{E2}$ are infinite, indicating that there is no need to limit the training epochs to produce optimal learning performance. Because there were no failed runs, the asymptotic performance was not modelled.

# 8  Conclusion

The measurement of learning performance in artificial neural networks is of great importance. Existing performance measurements have employed measures that are either dependent on an arbitrarily chosen training epoch limit or are strongly biased by the shortest learning times. By optimising the training epoch limit, I have developed new performance measures, the efficiency $e$ and the related mean $\bar{t}_E$, which are both independent of the training epoch limit and provide an unbiased measure of performance. The optimal training epoch limit $T_E$ and the range over which near-optimal performance is achieved $(T_{E1}, T_{E2})$ may be useful for solving novel problems.

I have also shown how the random distribution of learning times can be statistically mod-

elled, allowing prediction of the asymptotic success rate $\lambda$, and computation of corrected mean and median successful learning times, and I have demonstrated these new techniques on two popular benchmark problems. Further work is needed to extend the modelling to encompass a wider range of algorithms and to broaden the available base of benchmark results. In the process, it is believed that greater understanding of the learning processes of feed-forward artificial neural networks will result.

# References

M. S. Caceci and W. P. Cacheris. Fitting curves to data: The simplex algorithm is the answer. *Byte*, pages 340–362, May 1984.

Scott E. Fahlman. An empirical study of learning speed in back-propagation networks. Technical Report CMU-CS-88-162, Computer Science Department, Carnegie Mellon University, Pittsburgh, PA, 1988.

Leonard G. C. Hamey. Benchmarking feed-forward neural networks: Models and measures. Macquarie Computing Report, Computing Discipline, Macquarie University, NSW 2109 Australia, 1992.

R. A. Jacobs. Increased rates of convergence through learning rate adaptation. COINS Technical Report 87-117, University of Massachusetts at Amherst, Dept. of Computer and Information Science, Amherst, MA, 1987.

John F. Kolen and Jordan B. Pollack. Back propagation is sensitive to initial conditions. *Complex Systems*, 4:269–280, 1990.

John K. Kruschke and Javier R. Movellan. Benefits of gain: Speeded learning and minimal hidden layers in back-propagation networks. *IEEE Trans. Systems, Man and Cybernetics*, 21(1):273–280, January 1991.

Frederick Mosteller and John W. Tukey. *Data Analysis and Regression*. Addison-Wesley, 1977.

D. E. Rumelhart, G. E. Hinton, and R. J. Williams. Learning internal representations by error propagation. In *Parallel Distributed Processing*, chapter 8, pages 318–362. MIT Press, 1986.

M. A. Stephens. EDF statistics for goodness of fit and some comparisons. *Journal of the American Statistical Association*, 69:730–737, September 1974.

G. Tesauro and B. Janssens. Scaling relationships in back-propagation learning. *Complex Systems*, 2:39–44, 1988.

A. C. Veitch and G. Holmes. Benchmarking and fast learning in neural networks: Results for back-propagation. In *Proceedings of the Second Australian Conference on Neural Networks*, pages 167–171, 1991.

Yeong-Ho Yu and Robert F. Simmons. Descending epsilon in back-propagation: A technique for better generalization. In *Proceedings of the International Joint Conference on Neural Networks 1990*, 1990.
